# Bayesian model of surface perception

**William T. Freeman**
MERL, Mitsubishi Electric Res. Lab.
201 Broadway
Cambridge, MA 02139
freeman@merl.com

**Paul A. Viola**
Artificial Intelligence Lab
Massachusetts Institute of Technology
Cambridge, MA 02139
viola@ai.mit.edu

## Abstract

Image intensity variations can result from several different object surface effects, including shading from 3-dimensional relief of the object, or paint on the surface itself. An essential problem in vision, which people solve naturally, is to attribute the proper physical cause, e.g. surface relief or paint, to an observed image. We addressed this problem with an approach combining psychophysical and Bayesian computational methods.

We assessed human performance on a set of test images, and found that people made fairly consistent judgements of surface properties. Our computational model assigned simple prior probabilities to different relief or paint explanations for an image, and solved for the most probable interpretation in a Bayesian framework. The ratings of the test images by our algorithm compared surprisingly well with the mean ratings of our subjects.

## 1 Introduction

When people study a picture, they can judge whether it depicts a shaded, 3-dimensional surface, or simply a flat surface with markings or paint on it. The two images shown in Figure 1 illustrate this distinction [1]. To many observers Figure 1a appears to be a raised plateau lit from the left. Figure 1b is simply a re-arrangement of the local features of 1a, yet it does not give an impression of shape or depth. There is no simple correct answer for this problem; either of these images could be explained as marks on paper, or as illuminated shapes. Nevertheless people tend to make particular judgements of shape or reflectance. We seek an algorithm to arrive at those same judgements.

There are many reasons to study this problem. Disentangling shape and reflectance is a prototypical underdetermined vision problem, which biological vision systems routinely solve. Insights into this problem may apply to other vision problems

as well. A machine that could interpret images as people do would have many applications, such as the interactive editing and manipulation of images. Finally, there is a large body of computer vision work on "shape from shading"–inferring the 3-dimensional shape of a shaded object [4]. Virtually every algorithm assumes that all image intensity changes are caused by shading; these algorithms fail for any image with reflectance changes. To bring this body of work into practical use, we need to be able to disambiguate shading from reflectance changes.

There has been very little work on this problem. Sinha and Adelson [9] examined a world of painted polyhedra, and used consistancy constraints to identify regions of shape and reflectance changes. Their consistancy constraints involved specific assumptions which need not always hold and may be better described in a probabilistic framework. In addition, we seek a solution for more general, greyscale images.

Our approach combines psychophysics and computational modeling. First we will review the physics of image formation and describe the under-constrained surface perception problem. We then describe an experiment to measure the interpretations of surface shading and reflectance among different individuals. We will see that the judgements are fairly consistent across individuals and can be averaged to define "ground truth" for a set of test images. Our approach to modeling the human judgements is Bayesian. We begin by formulating prior probabilities for shapes and reflectance images, in the spirit of recent work on the statistical modeling of images [5, 8, 11]. Using these priors, the algorithm then determines whether an image is more likely to have been generated by a 3D shape or as a pattern of reflectance. We compare our algorithm's performance to that of the human subjects.

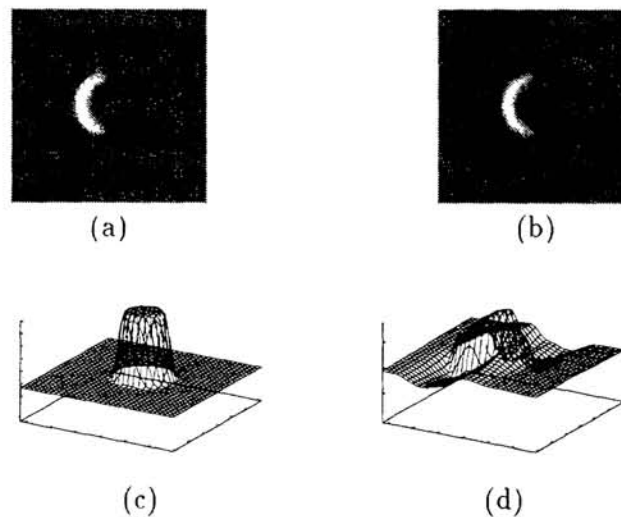

<center>(a)             (b)</center>

<center>(c)             (d)</center>

**Figure 1:** Images (a) and (b), designed by Adelson [1], are nearly the same everywhere, yet give different percepts of shading and reflectance. (a) looks like a plateau, lit from the left; (b) looks like marks on paper. Illustrating the under-constrained nature of perception, both images can be explained either by reflectance changes on paper (they are), or, under appropriate lighting conditions, by the shapes (c) and (d), respectively (vertical scale exaggerated).

## 2 Physics of Imaging

One simple model for the generation of an image from a three dimensional shape is the Lambertian model:

$$I(x,y) = R(x,y) \left( \hat{l} \cdot n(x,y) \right), \tag{1}$$

where $I(x,y)$ is an image indexed by pixel location, $\hat{n}(x,y)$ is the surface normal at every point on the surface conveniently indexed by the pixel to which that surface patch projects, $\hat{l}$ is a unit vector that points in the direction of the light source, and $R(x,y)$ is the reflectance at every point on the surface[1]. A patch of surface is brighter if the light shines onto it directly and darker if the light shines on it obliquely. A patch can also be dark simply because it is painted with a darker pigment. The shape of the object is probably more easily described as a depth map $z(x,y)$ from which $\hat{n}(x,y)$ is computed.

The classical "shape from shading" task attempts to compute $z$ from $I$ given knowledge of $\hat{l}$ and assuming $R$ is everywhere constant. Notice that the problem is "ill-posed"; while $I(x,y)$ does constrain $\hat{n}(x,y)$ it is not sufficient to uniquely determine the surface normal at each pixel. Some assumption about global properties of $z$ is necessary to *condition* the problem. If $R$ is allowed to vary, the problem becomes even more under-constrained. For example, $R = I$ and $\hat{n}(x,y) = \hat{l}$ is a valid solution for every image. This is the "all reflectance" hypothesis, where the inferred surface is flat and all of the image variation is due to reflectance. Interestingly there is also an "all shape" solution for every image where $R = 1$ and $I(x,y) = \hat{l} \cdot \hat{n}(x,y)$ (see Figure 1 for examples of such shapes).

Since the relationship between $z$ and $I$ is non-linear, "shape from shading" cannot be solved directly and requires a time consuming search procedure. For our computational experiments we seek a rendering model for shapes which simplifies the mathematics, yet maintains the essential ambiguities of the problem. We use the approximations of linear shading [6]. This involves two sets of approximations. First, that the rendered image $I(x,y)$ is some function, $G(\frac{\partial z}{\partial x}, \frac{\partial z}{\partial y})$, only of the surface slope at any point:

$$I(x,y) \approx G(\frac{\partial z}{\partial x}, \frac{\partial z}{\partial y}). \tag{2}$$

The second approximation is that the rendering function $G$ itself is a linear function of the surface slopes,

$$G(\frac{\partial z}{\partial x}, \frac{\partial z}{\partial y}) \approx k_1 + k_2 \frac{\partial z}{\partial x} + k_3 \frac{\partial z}{\partial y}. \tag{3}$$

Under linear shading, finding a shape which explains a given image is a trivial integration along the direction of the assumed light source. Despite this simplicity, images rendered under linear shading appear fairly realistically shaded [6].

## 3 Psychophysics

We used a survey to assess subjects' image judgements. We made a set of 60 test images, using Canvas and Photoshop programs to generate and manipulate the images. Our goal was to create a set of images with varying degrees of shadedness. We sought to assess to what extent each subject saw each image as created by

shading changes or reflectance changes. Each of our 18 naive observers was given a 4 page survey showing the images in a different random order.

To explain the problem of image interpretation quickly to naive subjects, we used a concrete story (Adelson's Theater Set Shop analogy [2] is a related didactic example). The survey instructions were as follows:

> *Pretend that each of the following pictures is a photograph of work made by either a painter or a sculptor.*
>
> *The painter could use paint, markers, air brushes, computer, etc., to make any kind of mark on a flat canvas. The paint had no 3-dimensionality; everything was perfectly flat.*
>
> *The sculptor could make 3-dimensional objects, but could make no markings on them. She could mold, sculpt, and scrape her sculptures, but could not draw or paint. All the objects were made out of a uniform plaster material and were made visible by lighting and shading effects.*

The subjects used a 5-point rating scale to indicate whether each image was made by the painter (P) or sculptor (S): S, S?, ?, P?, P.

## 3.1 Survey Results

We examined a non-parametric comparison of the image ratings, the rank order correlation (the linear correlation of image rankings in order of shapeness by each observer) [7]. Over all possible pairings of subjects, the rank order correlations ranged from 0.3 to 0.9, averaging 0.65. All of these correlations were statistically significant, most at the 0.0001 level. We concluded that for our set of test images, people do give a very similar set of interpretations of shading and reflectance.

We assigned a numerical value to each of the 5 survey responses (S=2; S?=1; ?=0; P?=-1; P=-2) and found the average numerical "shadedness" score for each image. Figure 2 shows a histogram of the survey responses for each image, ordered in decreasing order of shadedness. The two images of Figure 1 had average scores of 1.7 and -1.6, respectively, confirming the impressions of shading and reflectance. There was good consensus for the rankings of the most paint-like and most sculpture-like images; the middle images showed a higher score variance. The rankings by each individual showed a strong correlation with the rankings by the average of the remaining subjects, ranging from 0.6 to 0.9. Figure 4 shows the histogram of those correlations. The ordering of the images by the average of the subjects' responses provides a "ground truth" with which to compare the rankings of our algorithm. Figure 3, left, shows a randomly chosen subset of the sorted images, in decreasing order of assessed sculptureness.

## 4 Algorithm

We will assume that people are choosing the most probable interpretation of the observed image. We will adopt a Bayesian approach and calculate the most probable interpretation for each image under a particular set of prior probabilities for images and shapes. To parallel the choices we gave our subjects, we will choose between interpretations that account for the image entirely by shape changes, or entirely by reflectance changes. Thus, our images are either a rendered shape, multiplied by a uniform reflectance image, or a flat shape, multiplied by some non-uniform reflectance image.

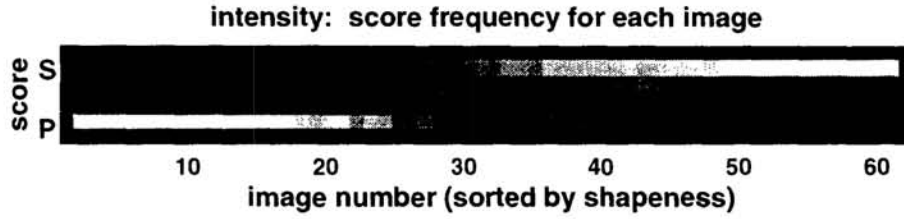

**Figure 2:** Histogram of survey responses. Intensity shows the number of responses of each score (vertical scale) for each image (horizontal, sorted in increasing order of shapeness).

To find the most probable interpretation, given an image, we need to assign prior probabilities to shape and reflectance configurations. There has been recent interest in characterizing the probabilities of images by the expected distributions of subband coefficient values [5, 8, 11]. The statistical distribution of bandpass linear filter outputs, for natural images, is highly kurtotic; the output is usually small, but in rare cases it takes on very large values. This non-gaussian behavior is not a property of the filter operation, because filtered "random" images appear gaussian. Rather it is a property of the structure of natural images. An exponential distribution, $P(c) \propto e^{-|c|}$, where $c$ is the filter coefficient value, is a reasonable model. These priors have been used in texture synthesis, noise removal, and receptive field modeling. Here, we apply them to the task of scene interpretation.

We explored using a very simple image prior:

$$P(I) \propto \exp\left(-\sum_{x,y} \sqrt{\frac{\partial I(x,y)}{\partial x}^2 + \frac{\partial I(x,y)}{\partial y}^2}\right) \quad (4)$$

Here we treat the image derivative as an image subband corresponding to a very simple filter. We applied this image prior to both reflectance images, $I(x,y)$, as well as range images, $z(x,y)$.

For any given picture, we seek to decide whether a shape or a reflectance explanation is more probable. The proper Bayesian approach would be to integrate the prior probabilities of all shapes which could explain the image in order to arrive at the total probability of a shape explanation. (The reflectance explanation, $\hat{R}$ is unique; the image itself). We employed a computationally simpler procedure, a very rough approximation to the proper calculation: we evaluated the prior probability, $P(\hat{S})$ of the single, most probable shape explanation, $\hat{S}$, for the image. Using the ratio test of a binary hypothesis, we formed a shapeness index, $J$, by the ratio of the probabilities for the shape and reflectance explanations, $J = \frac{P(\hat{S})}{P(\hat{R})}$. The index $J$ was used to rank the test images by shapeness.

We need to find the most probable shape explanation. The overall log likelihood of a shape, $z$, given an image is, using the linear shading approximations of Eq. (3):

$$
\begin{aligned}
\log P(z, k_1, k_2, k_3 | I) &= \log P(I|z, k_1, k_2, k_3) + \log P(z) + c \\
&= \sum_{x,y} (I - k_1 + k_2 \frac{\partial z}{\partial x} + k_3 \frac{\partial z}{\partial y})^2 + \sum_{x,y} \sqrt{\frac{\partial z}{\partial x}^2 + \frac{\partial z}{\partial y}^2} + c,
\end{aligned}
\quad (5)
$$

where $c$ is a normalization constant. We use a multi-scale gradient descent algorithm that simultaneously determines the optimal shape and illumination parameters for an image (similar to that used by [10]). The optimization procedure has three stages starting with a quarter resolution version of $I$, and moving to the half and

Human Rankings                              Algorithm Rankings

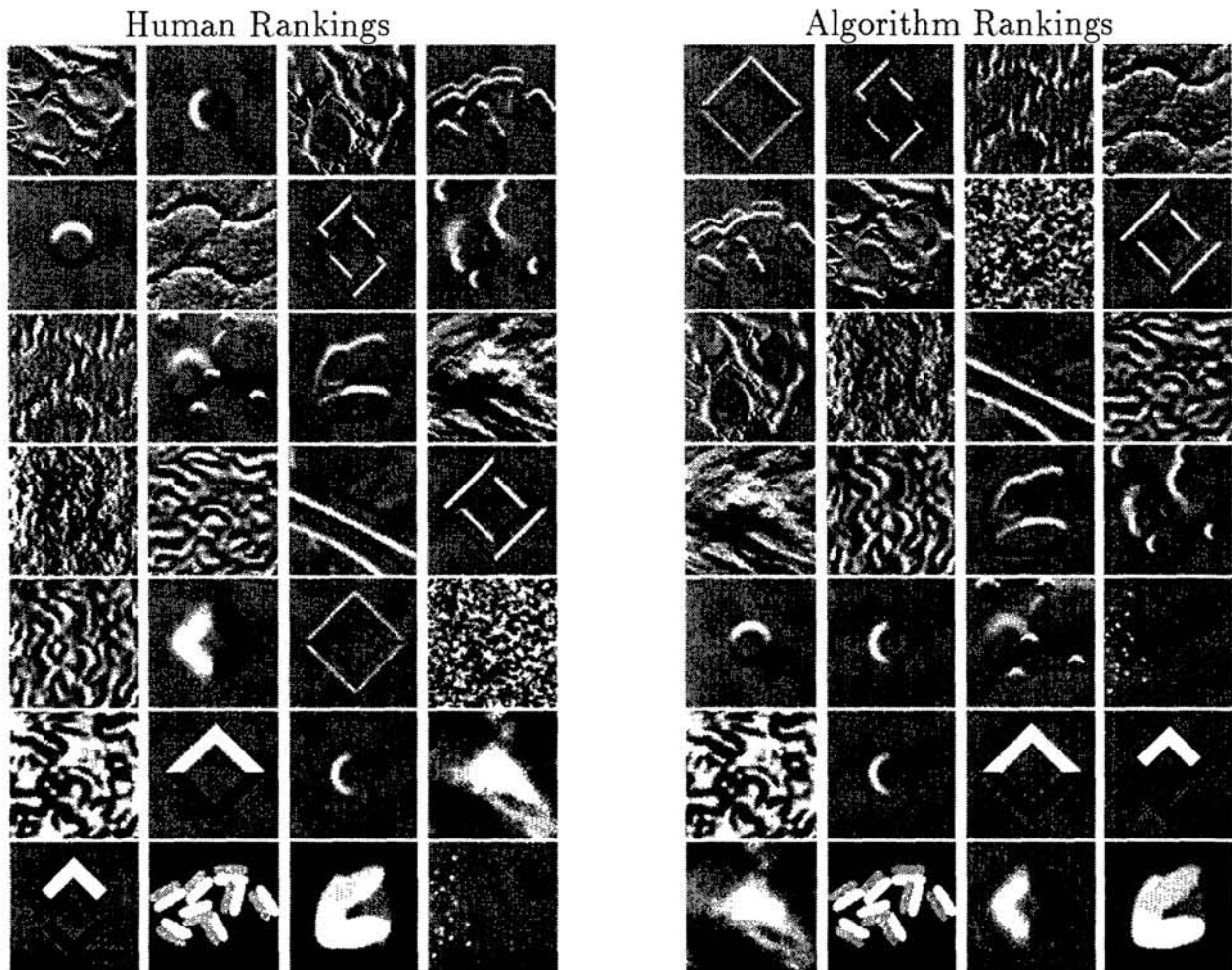

**Figure 3:** 28 of the 60 test images, arranged in decreasing order of subjects' shapeness ratings. Left: Subjects' rankings. Right: Algorithm's rankings.

then full resolution. The solution found at the low resolution is interpolated up to the next level and is used as a starting point for the next step in the optimization. In our experiments images are 128x128 pixels. The optimization procedure takes 4000 descent steps at each resolution level.

## 5   Results

Surprisingly, the simple prior probability of Eq. (4) accounts for much of the ratings of shading or paint by our human subjects. Figure 3 compares the rankings (shown in raster scan order) of a subset of the test images for our algorithm and the average of our subjects. The overall agreement is good. Figure 4 compares two measures: (1) the correlations (dark bars) of the subjects' individual ratings to the mean subject rating with (2) the correlation of our algorithm's ratings to the mean subject rating. Subjects show correlations between 0.6 and 0.9; our Bayesian algorithm showed a correlation of 0.64. Treating the mean subjects' ratings as the right answer, our algorithm did worse than most subjects but not as badly as some subjects.

Figure 1 illustrates how our algorithm chooses an interpretation for an image. If a simple shape explains an image, such as the shape explanation (c) for image (a), the shape gradient penalties will be small, assigning a high prior probability to that shape. If a complicated shape (d) is required to explain a simple image (b), the

low prior probability of the shape and the high prior probability of the reflectance image will favor a "paint" explanation.

We noted that many of the shapes inferred from paint-like images showed long ridges coincidently aligned with the assumed light direction. The assumption of generic light direction can be applied in a Bayesian framework [3] to penalize such coincidental alignments. We speculate that such a term would further penalize those unlikely shape interpretations and may improve algorithm performance.

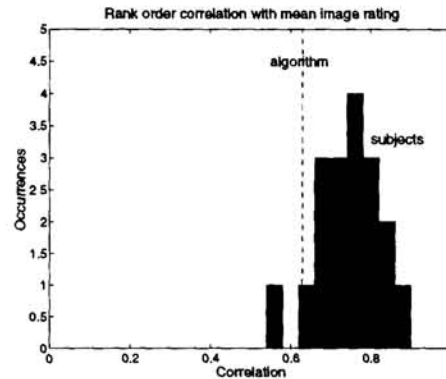

**Figure 4:** Correlation of individual subjects' image ratings compared with the mean rating (bars) compared with correlation of algorithm's rating with the mean rating (dashed line).

## Acknowledgements

We thank E. Adelson, D. Brainard, and J. Tenenbaum for helpful discussions.

## Footnotes

[1] Note: we assume orthographic projection, a distant light source, and no shadowing.

## References

[1] E. H. Adelson, 1995. personal communication.

[2] E. H. Adelson and A. P. Pentland. The perception of shading and reflectance. In B. Blum, editor, *Channels in the Visual Nervous System: Neurophysiology, Psychophysics, and Models*, pages 195–207. Freund Publishing, London, 1991.

[3] W. T. Freeman. The generic viewpoint assumption in a framework for visual perception. *Nature*, 368(6471):542–545, April 7 1994.

[4] B. K. P. Horn and M. J. Brooks. *Shape from shading*. MIT Press, Cambridge, MA, 1989.

[5] B. A. Olshausen and D. J. Field. Emergence of simple-cell receptive field properties by learning a sparse code for natural images. *Nature*, 381:607–609, 1996.

[6] A. P. Pentland. Linear shape from shading. *Intl. J. Comp. Vis.*, 1(4):153–162, 1990.

[7] W. H. Press, S. A. Teukolsky, W. T. Vetterling, and B. P. Flannery. *Numerical Recipes in C*. Cambridge Univ. Press, 1992.

[8] E. P. Simoncelli and E. H. Adelson. Noise removal via Bayesian wavelet coring. In *3rd Annual Intl. Conf. on Image Processing*, Laussanne, Switzerland, 1996. IEEE.

[9] P. Sinha and E. H. Adelson. Recovering reflectance and illumination in a world of painted polyhedra. In *Proc. 4th Intl. Conf. Computer Vision*, pages 156–163. IEEE, 1993.

[10] D. Terzopoulos. Multilevel computational processes for visual surface reconstruction. *Comp. Vis., Graphics, Image Proc.*, 24:52–96, 1983.

[11] S. C. Zhu and D. Mumford. Learning generic prior models for visual computation. Submitted to IEEE Trans. PAMI, 1997.